# Volume Regularization for Binary Classification

**Koby Crammer**
Department of Electrical Enginering
The Technion - Israel Institute of Technology
Haifa, 32000 Israel
koby@ee.technion.ac.il

**Tal Wagner**[*]
Faculty of Mathematics and Computer Science
Weizmann Institute of Science
Rehovot, 76100, Israel
tal.wagner@gmail.com

## Abstract

We introduce a large-volume box classification for binary prediction, which maintains a subset of weight vectors, and specifically axis-aligned boxes. Our learning algorithm seeks for a box of large volume that contains "simple" weight vectors which most of are accurate on the training set. Two versions of the learning process are cast as convex optimization problems, and it is shown how to solve them efficiently. The formulation yields a natural PAC-Bayesian performance bound and it is shown to minimize a quantity directly aligned with it. The algorithm outperforms SVM and the recently proposed AROW algorithm on a majority of 30 NLP datasets and binarized USPS optical character recognition datasets.

## 1 Introduction

Linear models are widely used for a variety of tasks including classification and regression. Support vectors machines [3, 22] (SVMs) are considered a primary method to efficiently build linear classifiers from data, yielding state-of-the-art performance. SVMs and many other methods are often easy to implement and efficient, yet return only a *single* weight-vector with no additional information about alternative models nor about confidence in prediction.

An alternative approach is taken by Bayesian methods [21, 13]. The primary object is a (posterior) distribution over models that is updated using Bayes rule. Unfortunately, the posterior is very complicated even for simple models, such as Bayesian logistic regression [15], and it is not known how to perform the update analytically, and approximations are required.

In this work we integrate the advantages of both approaches. We propose to model uncertainty over weight-vectors by maintaining a (simple) set of possible weight-vectors, rather than a single weight-vector. Learning is motivated from principles of discriminative learning rather than Bayes' rule, and it is optimizing a combination of an hand-crafted regularization term and the empirical loss. Specifically, our algorithm maintains an axis-aligned box, which only requires double number of parameters than maintaing a single weight-vector, a dominating model for many tasks.

We use a similar conceptual reasoning as used in Bayes point machines (BPM) [13]. Both approaches maintain a set of possible weights, which can be thought of as a posterior. BPMs use the version space, the set of all consistent weight vectors, which is a convex polyhedron. Since the size of the polyhedron's representation grows with the number of training examples, BPMs approximate the polyhedron with a single weight-vector, the Bayes point. Our algorithms model the set as a box, with a representation that is fixed in the size of the input, and find an optimal prediction box.

We cast learning as a convex optimization problem and propose methods to solve it efficiently. We further provide generalization bounds using PAC-Bayesian theory, and show that our algorithm is

---

[*]The research was performed while TW was a student at the Technion.

minimizing a quantity directly related to the generalization bound. We give two formulations or versions of the algorithm, one that is closely related to the bound, while the other is smooth.

We experiment with 30 binary text categorization datasets from various tasks: sentiment classification, predicting domain of product-review, assigning topics to news items, tagging spam emails, and classifying posts to news-groups. The results indicate that our algorithms outperform SVM and the recently proposed AROW [4] algorithm, which was shown to be the state-of-the-art in numerous NLP tasks. Additional support for the superiority and robustness of our algorithms, especially in high-noise setting, is provided using experiments with 45 pairs of binarized USPS OCR problems.

**Notation:** Given a vector $\boldsymbol{x} \in \mathbb{R}^d$, we denote its $k$th element by $\boldsymbol{x}_k \in \mathbb{R}$, and by $|\boldsymbol{x}| \in \mathbb{R}^d$ the vector with component-wise absolute value of its elements, $|\boldsymbol{x}| = (|\boldsymbol{x}_1|, \ldots, |\boldsymbol{x}_d|)$.

## 2    Large-Volume Box Classifiers

Standard linear classification learning algorithms maintain and return a single weight vector $\boldsymbol{w}^\star \in \mathbb{R}^d$ used to predict the label of any test point. We study a generalization of these algorithms where hypotheses are uncertainty (sub)sets of weight vectors $\boldsymbol{w}$. Such a hypothesis can be seen as a randomized linear classifier or a voting process. To classify an instance $\boldsymbol{x}$, a parameter vector $\boldsymbol{w}$ is drawn according to the hypothesis and predicts the label $\text{sign}(\boldsymbol{w} \cdot \boldsymbol{x})$. Herbrich et.al. [13, 12], argued in a similar context that such a randomization yields a more *robust* solution. PAC-Bayesian analysis and its generalization bounds give additional justification to this approach (see Sec 5).

The uncertainty subsets we study are axis aligned boxes parametrized with two vectors $\boldsymbol{u}, \boldsymbol{v} \in \mathbb{R}^d$ where we assume, $\boldsymbol{u}_k \leq \boldsymbol{v}_k$ for all $k = 1 \ldots d$. In words, $\boldsymbol{u}$ is the vertex with the lowest coordinates, and $\boldsymbol{v}$ is the vertex with the largest coordinates. The projection of the box onto the $k$-axis yields the interval $[\boldsymbol{u}_k, \boldsymbol{v}_k]$. The set of weight vectors contained in the box is denoted by, $\mathcal{Q} = \{\boldsymbol{w} : \boldsymbol{u}_k \leq \boldsymbol{w}_k \leq \boldsymbol{v}_k \text{ for } k = 1 \ldots d\}$. Given an instance $\boldsymbol{x}$ to be classified, a Gibbs classifier samples a weight vector uniformly in random from the box $\boldsymbol{w} \in \mathcal{Q}$ and returns $\text{sign}(\boldsymbol{w} \cdot \boldsymbol{x})$. A deterministic alternative we use in practice is to employ the center of mass defined by $\boldsymbol{\mu} = \frac{1}{2}(\boldsymbol{u} + \boldsymbol{v})$ and return $\text{sign}(\boldsymbol{\mu} \cdot \boldsymbol{x})$. For linear classifiers, the majority prediction with Gibbs sampling coincides with predicting using the center of mass. We also define the uncertainty intervals $\boldsymbol{\sigma} = \frac{1}{2}(\boldsymbol{v} - \boldsymbol{u})$. Intuitively, the uncertainty in the weight associated with the $k$th feature is $\boldsymbol{\sigma}_k$. Clearly, $\boldsymbol{v} = \boldsymbol{\mu} + \boldsymbol{\sigma}$ and $\boldsymbol{u} = \boldsymbol{\mu} - \boldsymbol{\sigma}$.

## 3    Learning as Optimization

Given a labeled sample $S = \{(\boldsymbol{x}_i, y_i)\}_{i=1}^n$, a common practice in learning linear models $\boldsymbol{w}$ is to perform structural risk minimization (SRM) [25] that picks a weight-vector that is both "simple" (eg small norm) and performs well on the training set. Learning is cast as an optimization problem,

$$\boldsymbol{w}^\star = \arg\min_{\boldsymbol{w}} \quad \frac{1}{n} \sum_i \ell(\boldsymbol{w}, (\boldsymbol{x}_i, y_i)) + D\,\mathcal{R}(\boldsymbol{w}) \,. \tag{1}$$

The first term is the empirical loss evaluated on the training set with some loss function $\ell(\boldsymbol{w}, (\boldsymbol{x}, y))$, and the second term is a *regularization* that penalizes weight-vectors according to their complexity. The parameter $D > 0$ is a tradeoff parameter.

Learning with uncertainty sets invites us to balance three desires rather than two as when learning a single weight-vector. The first two desires are generalizations of the structural risk minimization principle [25] to boxes: we prefer boxes containing weight-vectors that attain both low loss $\ell(\boldsymbol{w}, (\boldsymbol{x}_i, y_i))$ and are "simple" (eg small norm). This alone though is not enough, as if the loss and regularization functions are strictly convex then the optimal box would in fact be a single weight-vector. The third desire is thus to prefer boxes with large volume. Intuitively, if during training an algorithm finds a box with large volume, such that all weight-vectors belonging to it attain low training error and are simple, we expect the classifier based on the center of mass to be robust to noise or fluctuations. This will be formally stated in the analysis described in Sec 5. We formalize this requirement by adding a term that is inversely proportional to the volume of the box $\mathcal{Q}$.

We take a worst-case approach, and define the loss of the box $\mathcal{Q}$ given an example $(\boldsymbol{x}, y)$ denoted by $\ell(\mathcal{Q}, (\boldsymbol{x}, y))$ to be the loss of the worst member $\boldsymbol{w} \in \mathcal{Q}$. Similarly, we define the complexity of the box $\mathcal{Q}$ to be the complexity of the most complex member of the box $\boldsymbol{w} \in \mathcal{Q}$, formally, $\ell(\mathcal{Q}, (\boldsymbol{x}, y)) = \sup_{\boldsymbol{w} \in \mathcal{Q}} \ell(\boldsymbol{w}, (\boldsymbol{x}, y))$ and $\mathcal{R}(\mathcal{Q}) = \sup_{\boldsymbol{w} \in \mathcal{Q}} \mathcal{R}(\boldsymbol{w})$.

Putting it all together, we replace (1) with,

$$\mathcal{Q}^{\star} = \arg\min_{\mathcal{Q}\in\mathbf{Q}}\sup_{\boldsymbol{w}\in\mathcal{Q}}\left(\frac{1}{m}\sum_{i}\ell(\boldsymbol{w},(\boldsymbol{x}_i,y_i)) + D\mathcal{R}(\boldsymbol{w})\right), \qquad (2)$$

where $\mathbf{Q}$ is a set of boxes with some minimal volume. In other words, the algorithm is seeking for a set of alternative weight-vectors all of which are performing well on the training data. We expect this formulation to be robust, as a box is evaluated with its worst performing member.

We modify the problem by removing the constraint $\mathcal{Q} \in \mathbf{Q}$ and adding an equivalent penalty term to the objective, namely the log-volume of the box. We use the log-volume function for three reasons. First, it is a common barrier function in optimization [26], and in our case it keeps the box from actually shrinking to a zero volume box. Second, this choice is supported by the analysis below, and third, it is additive in the dimension of the data $d$, like all other quantities of the objective. Additionally, we bound the supremum over $\boldsymbol{w}$ with a sum of supremum operators. To conclude, we cast the learning problem as the following optimization problem over boxes,

$$\arg\min_{\mathcal{Q}}\frac{1}{m}\sum_{i}\sup_{\boldsymbol{w}\in\mathcal{Q}}\ell(\boldsymbol{w},(\boldsymbol{x}_i,y_i)) - C\log\operatorname{vol}\mathcal{Q} + D\sup_{\boldsymbol{w}\in\mathcal{Q}}\mathcal{R}(\boldsymbol{w}), \qquad (3)$$

where $C, D > 0$ are two trade-off parameters used to balance the three goals. (In the analysis below it will be shown that $D$ can be also interpreted as a Langrange multiplier of a constrained optimization problem.) We further develop the last equation by making additional assumptions over the loss function and the regularization. We assume that the loss is a *monotonically decreasing* function of the product $y(\boldsymbol{x}^{\top}\boldsymbol{w})$, often called the *margin* (or the *signed margin*). This is a property of many popular loss functions for binary classification, including the hinge-loss and its square used by SVMs [3, 22], exp-loss used by boosting [9], logistic-regression [11] and the Huber-loss [14]. Under this assumption we compute analytically the first term of the objective (3).

**Lemma 1** *If the loss function is monotonically decreasing in the margin,* $\ell(\boldsymbol{w},(\boldsymbol{x},y)) = \ell(y(\boldsymbol{x}^{\top}\boldsymbol{w}))$ *then* $\sup_{\boldsymbol{w}\in\mathcal{Q}}\ell(\boldsymbol{w},(\boldsymbol{x}_i,y_i)) = \ell(y(\boldsymbol{x}^{\top}\boldsymbol{\mu}) - |\boldsymbol{x}|\boldsymbol{\sigma})$.

**Proof:** From the monotonicity of $\ell(\cdot)$ we have $\sup_{\boldsymbol{w}\in\mathcal{Q}}\ell\left(y(\boldsymbol{x}^{\top}\boldsymbol{w})\right) = \ell\left(\inf_{\boldsymbol{w}\in\mathcal{Q}}y(\boldsymbol{x}^{\top}\boldsymbol{w})\right)$. Computing the infimum we get,

$$\inf_{\boldsymbol{w}\in\mathcal{Q}}y(\boldsymbol{x}^{\top}\boldsymbol{w}) = \inf_{\boldsymbol{w}_k\in[\boldsymbol{u}_k,\boldsymbol{v}_k]\text{ for }k=1\ldots d}\sum_{k=1}^{d}(y\boldsymbol{x}_k)\boldsymbol{w}_k = \sum_{k=1}^{d}\inf_{\boldsymbol{w}_k\in[\boldsymbol{u}_k,\boldsymbol{v}_k]}(y\boldsymbol{x}_k)\boldsymbol{w}_k$$

$$= \sum_{k=1}^{d}(y\boldsymbol{x}_k)\left\{\begin{array}{ll}\boldsymbol{u}_k & (y\boldsymbol{x}_k)\geq 0\\ \boldsymbol{v}_k & (y\boldsymbol{x}_k)< 0\end{array}\right. = \sum_{k=1}^{d}(y\boldsymbol{x}_k)\left(\boldsymbol{\mu}_k - \operatorname{sign}(y\boldsymbol{x}_k)\boldsymbol{\sigma}_k\right) = y(\boldsymbol{x}^{\top}\boldsymbol{\mu}) - |\boldsymbol{x}|\boldsymbol{\sigma},$$

using $\boldsymbol{u} = \boldsymbol{\mu} - \boldsymbol{\sigma}$ and $\boldsymbol{v} = \boldsymbol{u} + \boldsymbol{\sigma}$ as stated above. ∎

The lemma highlights the need to constrain the volume to be strictly larger than zero: due to monotonicity and the fact that $\boldsymbol{\sigma} \geq 0$ (component wise) we have $\ell(y(\boldsymbol{x}^{\top}\boldsymbol{\mu}) - |\boldsymbol{x}|\boldsymbol{\sigma}) \geq \ell(y(\boldsymbol{x}^{\top}\boldsymbol{\mu}))$, so the loss is always minimized when we set $\boldsymbol{\sigma} = 0$. We next turn to analyse the third term of (3) with the following lemma.

**Lemma 2** *(1) Assuming $\mathcal{R}(\boldsymbol{w})$ is convex, then $\sup_{\boldsymbol{w}\in\mathcal{Q}}\mathcal{R}(\boldsymbol{w})$ is attained on* vertices *of the box $\mathcal{Q}$. (2) Additionally, if $\mathcal{R}(\boldsymbol{w})$ is* strictly *convex then the supremum is attained only on vertices.*

**Proof:** We use the fact that every point in the box can be represented as a convex combination of the vertices. Formally, given a point in the box $\boldsymbol{w} \in \mathcal{Q}$, there exists a vector $\boldsymbol{\alpha} \in \mathbb{R}^{2^d}$ with non-negative elements and $\sum_t \boldsymbol{\alpha}_t = 1$ such that $\boldsymbol{w} = \sum_t \boldsymbol{\alpha}_t \boldsymbol{z}_t$ where $\boldsymbol{z}_t$ are the vertices of the box. Convexity of $\mathcal{R}(\cdot)$ yields, $\mathcal{R}(\boldsymbol{w}) \leq \sum_t \boldsymbol{\alpha}_t \mathcal{R}(\boldsymbol{z}_t) \leq \max_t \{\mathcal{R}(\boldsymbol{z}_t)\}$. Thus, if $\boldsymbol{w}$ attains the supremum $\sup_{\boldsymbol{w}\in\mathcal{Q}}\mathcal{R}(\boldsymbol{w})$ then so does at least one vertex. Additionally, if $\mathcal{R}(\boldsymbol{w})$ is a strictly convex function, then the first inequality in the last equation is a strict inequality, and thus a non-vertex cannot attain the supremum. ∎

Common regularization functions are defined as sums over individual features, that is $\mathcal{R}(\boldsymbol{w}) = \sum_k r(\boldsymbol{w}_k)$. In this case the supremum is attained on each coordinate independently as follows.

**Corollary 3** *Assuming $\mathcal{R}(\boldsymbol{w})$ is a sum of scalar-convex functions $\sum_k r(\boldsymbol{w}_k)$, we have,*

$$\sup_{\boldsymbol{w} \in \mathcal{Q}} \mathcal{R}(\boldsymbol{w}) = \sum_k \max \{r(\boldsymbol{u}_k), r(\boldsymbol{v}_k)\} \qquad = \sum_k \max \{r(\boldsymbol{\mu}_k - \boldsymbol{\sigma}_k), r(\boldsymbol{\mu}_k + \boldsymbol{\sigma}_k)\} \ .$$

The corollary follows from the lemma since a supremum of a scalar-function over a box is equivalent to taking the supremum over the box projected to a single coordinate. Finally, the volume of a box is given by a product of the length of its axes, that is, $\text{vol}\,(\mathcal{Q}) = \prod_k (\boldsymbol{v}_k - \boldsymbol{u}_k) = \prod_k (2\boldsymbol{\sigma}_k) = 2^d \prod_k \boldsymbol{\sigma}_k$ .

To summarize, the learning problem of the large-volume box algorithm is cast by solving the following minimization problem, in terms of the center $\boldsymbol{\mu}$ and the size (or dimensions) $\boldsymbol{\sigma}$,

$$\min_{\boldsymbol{\sigma} \geq 0, \boldsymbol{\mu}} \frac{1}{m} \sum_{i=1}^{m} \ell \left( y_i(\boldsymbol{x}_i^\top \boldsymbol{\mu}) - |\boldsymbol{x}_i|\boldsymbol{\sigma}) \right) - C \sum_k \log \boldsymbol{\sigma}_k + D \sum_k \max \{r(\boldsymbol{\mu}_k - \boldsymbol{\sigma}_k), r(\boldsymbol{\mu}_k + \boldsymbol{\sigma}_k)\} \,, \quad (4)$$

where $\ell(\cdot)$ is a monotonically decreasing function, $r(\cdot)$ is a convex function, and $C, D > 0$ are two trade-off parameters used to balance our three desires. We denote by

$$\boldsymbol{z}_{i,+} = y_i\boldsymbol{x}_i + |\boldsymbol{x}_i| \in \mathbb{R}^d \,, \ \boldsymbol{z}_{i,-} = y_i\boldsymbol{x}_i - |\boldsymbol{x}_i| \in \mathbb{R}^d \ . \quad (5)$$

The k*th* element of $\boldsymbol{z}_{i,+}$ ($\boldsymbol{z}_{i,-}$) is twice the k*th* element of $|\boldsymbol{x}_i|$ if the sign of the k*th* element of $\boldsymbol{x}_i$ agrees (disagrees) with $y_i$, and zero otherwise.

This problem can equivalently be written in terms of the two "extreme" vertices $\boldsymbol{u}$ and $\boldsymbol{v}$ as follows,

$$\min_{\boldsymbol{v} \geq \boldsymbol{u}} \frac{1}{m} \sum_{i=1}^{m} \ell \left( \frac{1}{2} \left( \boldsymbol{v}^\top(\boldsymbol{z}_{i,-}) + \boldsymbol{u}^\top(\boldsymbol{z}_{i,+}) \right) \right)$$
$$- C \sum_k \log (\boldsymbol{v}_k - \boldsymbol{u}_k) + D \sum_k \max \{r(\boldsymbol{v}_k), r(\boldsymbol{u}_k)\} \,, \quad (6)$$

by using the relation $y_i\boldsymbol{x}_i^\top(\boldsymbol{v}+\boldsymbol{u}) - |\boldsymbol{x}_i|(\boldsymbol{v}-\boldsymbol{u}) = \boldsymbol{v}^\top(\boldsymbol{z}_{i,-}) + \boldsymbol{u}^\top(\boldsymbol{z}_{i,+})$ . Note, if the loss function $\ell(\cdot)$ is convex, then both formulations (4) and (6) of the learning problem are convex in their arguments, as each is a sum of convex functions of linear combination of the arguments, and a maximum of convex functions is convex.

We conclude this section with an additional alternative formulation, which for convenience, we present in the notation of (6). Although the above problem is convex, the regularization term $\sum_k \max \{r(\boldsymbol{v}_k), r(\boldsymbol{u}_k)\}$ is not smooth because of the $\max$ operator. In this alternative, we replace it with a smooth term, by changing the $\max$ to a sum, yielding $\sum_k r(\boldsymbol{v}_k) + r(\boldsymbol{u}_k) = \mathcal{R}(\boldsymbol{u}) + \mathcal{R}(\boldsymbol{v})$. The problem then becomes,

$$\min_{\boldsymbol{v} \geq \boldsymbol{u}} \frac{1}{m} \sum_{i=1}^{m} \ell \left( \frac{1}{2} \left( \boldsymbol{v}^\top(\boldsymbol{z}_{i,-}) + \boldsymbol{u}^\top(\boldsymbol{z}_{i,+}) \right) \right) - C \sum_k \log (\boldsymbol{v}_k - \boldsymbol{u}_k) + D \left( \mathcal{R}(\boldsymbol{u}) + \mathcal{R}(\boldsymbol{v}) \right) \ . \quad (7)$$

The two alternatives are related via the following chain of inequalities, $0.5 \max \{r(\boldsymbol{v}_k), r(\boldsymbol{u}_k)\} \leq 0.5 \left( r(\boldsymbol{v}_k) + r(\boldsymbol{u}_k) \right) \leq \max \{r(\boldsymbol{v}_k), r(\boldsymbol{u}_k)\} \leq r(\boldsymbol{v}_k) + r(\boldsymbol{u}_k)$ . In other words, given either one of the problems (6) or (7), we can lower and upper bound it with the other problem with a proper choice of trade-off parameter $D$. We call the two versions BoW for box-of-weights algorithm, and refer to them as BoW-M(ax) and BoW-S(um), respectively.

## 4 Optimization Algorithm

We now present an algorithm to solve (6) for the special case $r(x) = x^2$. The algorithm is a based on COMID [8] and its convergence analysis follows directly from the analysis of COMID, which is omitted due to lack of space. The algorithm works in iterations. On each iteration a (stochastic) gradient decent step is performed, followed by a regularization-optimization step. Formally, the algorithm picks a random example $i$ and updates,

$$(\tilde{\boldsymbol{u}}, \tilde{\boldsymbol{v}}) \leftarrow (\boldsymbol{u}, \boldsymbol{v}) - \alpha \frac{\eta}{2} \left( \boldsymbol{z}_{i,+} \,, \, \boldsymbol{z}_{i,-} \right) \ \text{for} \ \alpha = \ell' \left( \frac{1}{2} \left( \boldsymbol{v}^\top(\boldsymbol{z}_{i,-}) + \boldsymbol{u}^\top(\boldsymbol{z}_{i,+}) \right) \right) \ .$$

The algorithm then solves the following regularization-oriented optimization problem,

$$\min_{\boldsymbol{u},\boldsymbol{v}} \frac{1}{2}\|\boldsymbol{u} - \tilde{\boldsymbol{u}}\|^2 + \frac{1}{2}\|\boldsymbol{v} - \tilde{\boldsymbol{v}}\|^2 - C\sum_k \log\left(\boldsymbol{v}_k - \boldsymbol{u}_k\right) + D\sum_k \max\left\{\boldsymbol{v}_k^2, \boldsymbol{u}_k^2\right\} \ .$$

The objective of the last problem decomposes over individual pairs $\boldsymbol{u}_k, \boldsymbol{v}_k$ so we reduce the optimization to $d$ independent problems, each defined over 2 scalars $u$ and $v$ (omitting index $k$),

$$\min_{u,v} F(u,v) = \frac{1}{2}\left(u - \tilde{u}\right)^2 + \frac{1}{2}\left(v - \tilde{v}\right)^2 - C\log\left(v - u\right) + D\max\left\{v^2, u^2\right\} \ . \qquad (8)$$

We denote the half-plane $H = \{(u,v) \in \mathbb{R}^2 : v > u\}$ and partition it into three subsets: $G_1 = \{(u,v) \in H : v > -u\}$, $G_2 = \{(u,v) \in H : v < -u\}$, and the line $L = \{(u,v) \in \mathbb{R}^2 : v = -u\}$. The following lemma describes the optimal solution of (8).

**Lemma 4** *Exactly one of the items below holds and describe the optimal solution of* (8).

1. *If there exists $(u,v) \in G_1$ such that $v$ is a root of $f(v) = \alpha v^2 + \beta v + \gamma$ and $u = \tilde{u} - 2Dv + (\tilde{v} - v)$ where $\alpha = 2(1 + D)(1 + 2D)$, $\beta = -\tilde{u}(1 + 2D) - \tilde{v}(3 + 4D)$, and $\gamma = \tilde{v} + \tilde{u}\tilde{v} - C$, then it is a global minimum of F.*

2. *If there exists $(u,v) \in G_2$ such that $u$ is a root of $f(u) = \alpha u^2 + \beta u + \gamma$ and $v = \tilde{v} - 2Du + (\tilde{u} - u)$ where $\alpha = 2(1 + D)(1 + 2D)$, $\beta = -\tilde{v}(1 + 2D) - \tilde{u}(3 + 4D)$ and $\gamma = \tilde{u} + \tilde{v}\tilde{u} - C$, then it is a global minimum of F. Furthermore, such point and a point described in 1 cannot exist simultaneously.*

3. *If no points as described in 1 nor 2 exist, then the global minimum of F is $(u, -u)$ such that $u$ is a root of $f(u) = \alpha u^2 + \beta u + C$ where $\alpha = 2 + 2D$, $\beta = \tilde{v} - \tilde{u}$, $\gamma = -C$.*

**Proof sketch:** By definition, the function $F$ is smooth and convex on $G_1$. The condition in 1 is equivalent to satisfying $\nabla F(u,v) = 0$, and therefore any point that satisfies it, is a minimum of $F\big|_{G_1}$. A similar argument applies to $G_2$ with 2. The convexity of $F$ on the entire set $H$ yields that any such point is also a global minimum of $F$, and that if no such point exists then $F$ attains a global minimum on $L$ (which is derived in 3). The latter is sure to exist since $\lim_{v \to 0} F|_L = \lim_{v \to \infty} F|_L = \infty$. The algebraic derivation is omitted due to lack of space. ∎

Similarly, we develop the update for solving (7). Here after the gradient step we need to solve the following problem per coordinate $k$, $\min_{u,v} F(u,v) = \frac{1}{2}\left(u - \tilde{u}\right)^2 + \frac{1}{2}\left(v - \tilde{v}\right)^2 - C\log\left(v - u\right) + D\left(v^2 + u^2\right)$ . The following lemma characterizes the optimal solution.

**Lemma 5** *The optimal solution $(u,v) \in \{(u,v) \in \mathbb{R}^2 : v - u > 0\}$ of the last problem is such that $u$ is a root of the polynomial $f(u) = \alpha u^2 + \beta u + \gamma$ where $\alpha = 2 + 2D + 6D + 8D^2$, $\beta = -(\tilde{v} + 2D\tilde{v} + \tilde{u} + 6D\tilde{u}) - 2\tilde{u}$, $\gamma = \tilde{u}^2 + \tilde{u}\tilde{v} - 4C - 2CD$ and $v = (\tilde{v} + \tilde{u} - u(1 + 2D))/(1 + 2D)$.*

Its proof is similar to the proof of Lemma 4, but simpler and omitted due to lack of space.

## 5 Analysis

PAC-Bayesian bounds were introduced by McAllester [19], were further refined later (e.g. [17, 23]), and applied to analyze SVMs [18]. They often have been shown to be quite tight.

We first introduce some notation needed for the discussion of these bounds. Let $\bar{\ell}(\boldsymbol{w}, (\boldsymbol{x}, y))$ denote the zero-one loss, that is $\bar{\ell}(\boldsymbol{w}, (\boldsymbol{x}, y)) = 1$ if $\mathrm{sign}(\boldsymbol{w} \cdot \boldsymbol{x}) \neq y$ and $\bar{\ell}(\boldsymbol{w}, (\boldsymbol{x}, y)) = 0$ otherwise. Let $\mathcal{D}$ be a distribution over the labeled examples $(\boldsymbol{x}, y)$, and denote by $\bar{\ell}(\boldsymbol{w}, \mathcal{D})$ the expected zero-one loss of a linear classifier characterized by its weight vector $\boldsymbol{w}$: $\bar{\ell}(\boldsymbol{w}, \mathcal{D}) = \mathrm{Pr}_{(\boldsymbol{x},y)\sim\mathcal{D}}[\mathrm{sign}(\boldsymbol{w} \cdot \boldsymbol{x}) \neq y] = \mathrm{E}_{(\boldsymbol{x},y)\sim\mathcal{D}}[\bar{\ell}(\boldsymbol{w}, (\boldsymbol{x}, y))]$ . We abuse notation, and denote by $\bar{\ell}(\boldsymbol{w}, S)$ the expected loss $\bar{\ell}(\boldsymbol{w}, \mathcal{D}_S)$ for the empirical distribution $\mathcal{D}_S$ of a sample $S$.

PAC-Bayesian analysis states generalization bounds in terms of two distributions - prior and posterior - over all hypotheses (i.e. over weight-vectors $\boldsymbol{w}$). Below, we identify a compact set with a uniform distribution over the set, and in particular we identify a box $\mathcal{Q}$ with a uniform distribution

over all weight vectors it contains (and zero mass otherwise). Similarly, we identify any compact body $\mathcal{P}$ with a uniform distribution over its elements. In other words, we refer to the prior $\mathcal{P}$ and the posterior $\mathcal{Q}$ both as two uniform distributions and as their support (which are subsets). We also denote by $\ell(\mathcal{Q}, \mathcal{D})$ the expectation of $\ell(\boldsymbol{w}, \mathcal{D})$ over weight vectors $\boldsymbol{w}$ drawn according to the distribution $\mathcal{Q}$. We quote Cor. 2.2 of Germain et.al. [10],

**Corollary 6 ([10])** : *For any distribution $\mathcal{D}$, for any set $\mathcal{H}$ of weight-vectors, for any distribution $\mathcal{P}$ of support $\mathcal{H}$, for any $\delta \in (0, 1]$, and any positive number $\gamma$ the following statement holds with probability $\geq 1 - \delta$ over samples $S$ of size $n$,*

$$\bar{\ell}(\mathcal{Q}, \mathcal{D}) \leq \frac{1}{1 - e^{-\gamma}} \left\{ 1 - \exp\left[ -\left( \gamma \cdot \bar{\ell}(\mathcal{Q}, S) + \frac{1}{n} D_{KL}(\mathcal{Q}\|\mathcal{P}) + \frac{1}{n} \ln\left(\frac{1}{\delta}\right) \right) \right] \right\} . \quad (9)$$

The corollary states that the expected number of mistakes over examples drawn according to some fixed and unknown distribution $\mathcal{D}$ over inputs, and over weight-vectors drawn from the box $\mathcal{Q}$ uniformly, is bounded by the right term, which is a monotonic function of the following sum,

$$\bar{\ell}(\mathcal{Q}, S) + \frac{1}{n\gamma} D_{KL}(\mathcal{Q}\|\mathcal{P}) . \quad (10)$$

For uniform distributions we have the following,

$$D_{KL}(\mathcal{Q}\|\mathcal{P}) = \begin{cases} \log \frac{\text{vol}(\mathcal{P})}{\text{vol}(\mathcal{Q})} & \mathcal{Q} \subseteq \mathcal{P} \\ \infty & \text{otherwise} \end{cases} . \quad (11)$$

Additionally, we bound the empirical training error,

$$\bar{\ell}(\mathcal{Q}, S) = \frac{1}{n} \sum_i^n \frac{1}{\text{vol}\mathcal{Q}} \int_{\boldsymbol{w}\in\mathcal{Q}} \bar{\ell}(\boldsymbol{w}, (\boldsymbol{x}_i, y_i))\, d\boldsymbol{w} \leq \frac{1}{n} \sum_i \ell\left( \inf_{\boldsymbol{w}\in\mathcal{Q}} y_i(\boldsymbol{x}_i^\top \boldsymbol{w}) \right), \quad (12)$$

where the equality is the definition of $\bar{\ell}(\mathcal{Q}, S)$, and the inequality follows by choosing a loss function $\ell(\cdot)$ which upper bounds the zero-one loss (e.g. Hinge loss), by bounding an expectation with the supremum value, and from Lemma 1.

We get that to minimize the generalization bound of (9) we can minimize a bound on (10) which is obtained by substituting (11) and (12) in (10). Omitting constants we get,

$$\min_{\mathcal{Q}} \frac{1}{n} \sum_i \ell\left( \inf_{\boldsymbol{w}\in\mathcal{Q}} y_i \boldsymbol{w}^\top \boldsymbol{x}_i \right) - \frac{1}{n\gamma} \log \text{vol}\mathcal{Q} \quad \text{s.t. } \mathcal{Q} \subseteq \mathcal{P} . \quad (13)$$

Next, we set $\mathcal{P}$ to be a ball of radius $R$ about the origin, and, as in Sec 2, we set $\mathcal{Q}$ as a box parametrized with the vectors $\boldsymbol{u}$ and $\boldsymbol{v}$. We use the following lemma, of which proof is omitted due to lack of space,

**Lemma 7** *If $\mathcal{P}$ is a ball of radius $R$ about the origin and $\mathcal{Q}$ is a box parametrized using $\boldsymbol{u}$ and $\boldsymbol{v}$, we have $\mathcal{Q} \subseteq \mathcal{P} \Leftrightarrow \sum_k \max\{\boldsymbol{v}_k^2, \boldsymbol{u}_k^2\} \leq R^2$ .*

Finally, plugging Lemma 7 and Lemma 1 in (13) we get the following problem, which is monotonically related to a bound of the generalization loss, $\min_{\boldsymbol{v}\geq\boldsymbol{u}} \frac{1}{n} \sum_{i=1}^m \ell\left( \frac{1}{2}\left( \boldsymbol{v}^\top(\boldsymbol{z}_{i,-}) + \boldsymbol{u}^\top(\boldsymbol{z}_{i,+}) \right) \right) - \frac{1}{n\gamma} \sum_k \log(\boldsymbol{v}_k - \boldsymbol{u}_k)$ subject to $\sum_k \max\{r(\boldsymbol{v}_k), r(\boldsymbol{u}_k)\} \leq R^2$ . To solve the last problem we write its Lagrangian,

$$\max_{\eta} \min_{\boldsymbol{v}\geq\boldsymbol{u}} \frac{1}{n} \sum_{i=1}^m \ell\left( \frac{1}{2}\left( \boldsymbol{v}^\top(\boldsymbol{z}_{i,-}) + \boldsymbol{u}^\top(\boldsymbol{z}_{i,+}) \right) \right) - \frac{1}{n\gamma} \sum_k \log(\boldsymbol{v}_k - \boldsymbol{u}_k)$$
$$+ \eta \sum_k \max\{r(\boldsymbol{v}_k), r(\boldsymbol{u}_k)\} - \eta R^2 , \quad (14)$$

where $\eta$ is the Lagrange multiplier ensuring the constraint. Comparing (14), whose objective is used in the PAC-Bayesian bound, and our learning algorithm in (6), we observe that the three terms in both objectives are the same by setting $C = \frac{1}{n\gamma}$ and identifying the optimal value of the Lagrange

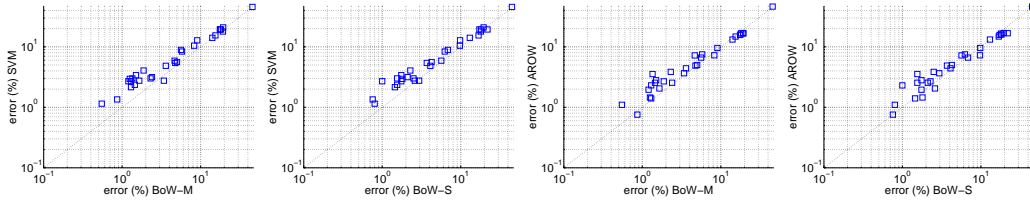

Figure 1: Fraction of error on text classification datasets of BoW-M and BoW-S vs SVM (two left plots); and BoW-M and BoW-S vs AROW (two right plots). Markers above the line indicate superior BoW performance.

multipler with the trade-off constant $\eta = D$. In fact, each value of the radius $R$ yields a unique optimal value of the Lagrange multiplier $\eta$. Thus, we can interpret the role of the constant $D$ as setting implicitly the effective radius of the *prior* ball $\mathcal{P}$.

Few comments are in order. First, the KL-divergence between distributions is minimized more effectively if both $\mathcal{P}$ and $\mathcal{Q}$ are of the same form, e.g. both $\mathcal{P}$ and $\mathcal{Q}$ are boxes. However, we chose $\mathcal{Q}$ to be a box, as it has a nice interpretation of uncertainty over features, and $\mathcal{P}$ to be a ball, as it decomposes (as opposed to an $\ell_\infty$ ball), which allows simpler optimization algorithms. Second, as noted above, BoW-S is indeed smoother than BoW-M, yet, from (14) it follows that the latter is better motivated from the PAC-Bayesian bound, as we want $\mathcal{Q} \subseteq \mathcal{P}$. Third, the bound is small if the volume of the box $\mathcal{Q}$ is large, which motivates seeking for large-volume boxes, whose members perform well.

## 6    Empirical Evaluation

We evaluated BoW-M and BoW-S on NLP tasks experimenting with all the $12$ datasets used by Dredze et al [6] (sentiment classification in 6 Amazon domains, 3 pairs of 20 newsgroups and 3 pairs of Reuters (RCV1)). We defined an additional task from the 6 Amazon domains (book, dvd, music, video, electronics, kitchen). Given reviews from two domains, the goal is to identify the domain identity. We used all $6 \times 5/2 = 15$ unordered pairs of domains. Additionally, we selected 3 users from task A of the 2006 ECML/PKDD Discovery Challenge spam data set. The goal is to classify an email as either a spam or a not-spam. This yielded a total of 30 datasets. For each problem we selected about $2,000$ instances and represented them with vectors of uni/bi-grams counts. Feature extraction followed a previous protocol [6, 2]. Each dataset was randomly divided for 10-fold cross validation. We also experimented with USPS OCR data which we binarized into 45 all-pairs problems, maintaining the standard split into training and test sets. Given an image of one of two digits, the goal is to detect which of the two digits is shown in the image.

We implemented BoW-M and BoW-S both with Hinge loss and Huber loss. The performance of the latter was slightly worse than the former, thus we report results only for the Hinge loss. We also tried AdaGrad [7] but surprisingly it did not work as well as COMID. We compared BoW with support vector machines (SVM) [3] and AROW [4] which was shown to outperform many algorithms on NLP tasks. (Other algorithms we evaluated, including maximum-entropy and SGD with Huber-loss, performed worse than either of these two algorithms and thus are omitted.) It is not clear at this point how to incorporate Mercer kernels into BoW, and thus we are restricted to evaluate all algorithms on data that can be classified well with linear models.

Classifiers parameters ($C$ for SVM, $r$ for AROW and $C, D$ for BoW) were tuned for each task on a single additional randomized run over the data splitting it into $80\%$, used for training, and the remaining $20\%$ of examples were used to choose the parameters. Results are reported for NLP tasks as the average error over the 10 folds per problem , while for USPS the standard test sets are used.

The mean error for 30 NLP tasks over 10 folds of BoW-M and BoW-S vs SVM is summarized in the two left panels of Fig. 1. Markers above the line indicate superior BoW performance. Clearly, both BoW versions outperform SVM obtaining lower test error on most (26) datasets and higher only on few (at most 3). The right two panels compare the performance of both BoW versions with AROW. Here the trend remains yet with a smaller gap, BoW-M outperforms AROW in 20 datasets, and is outperformed in 9, while BoW-S outperforms AROW in 19 datasets and outperformed in 12. Note, AROW was previously shown [4] to have superior performance on text data over other algorithms.

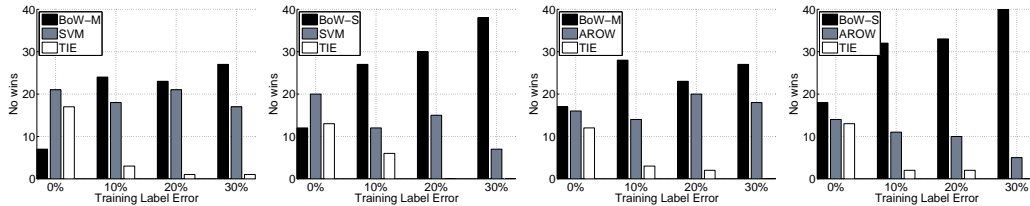

Figure 2: No. of USPS 1-vs-1 datasets (out of 45) for which one algorithm is better than the other (see legend) shown for four levels of label noise during training: $0\%, 10\%, 20\%$ and $30\%$ (left to right). Higher values indicate better performance.

The results of the experiments with USPS are summarized in Fig. 2. Each panel shows the number of datasets (out of 45) for which one algorithm outperforms another algorithm, for four levels of label noise (i.e. probability of flipping the correct label) during training: $0\%, 10\%, 20\%$ and $30\%$. The four pairs compared are BoW vs SVM (two left panels, BoW-M most left) and BoW vs AROW (two right panels, BoW-S most right). A left bar higher than a middle bar (in each group in each panel) indicates superior BoW performance. With no label noise (left group in each panel) SVM outperforms both BoW algorithms (e.g. SVM attains lower test error than BoW-S on 20 datasets and higher on 12 datasets, with a tie in 13 datasets). The average test error of SVM is 1.81, AROW is 1.98 and BoW-S is 1.97. When the level of noise increases both BoW algorithms outperform AROW and SVM. With maximal level of $30\%$ label noise, the average test error is $16.1\%$ for SVM, 14.8 for AROW, and $6.1\%$ for BoW-S. BoW-M achieves lower test error on 27 datasets (compared both with SVM and AROW), while BoW-S achieves lower test error than SVM on 38 datasets and than AROW on 40 datasets. Interestingly, while, in general, BoW-M achieved lower test error than BoW-S on the NLP problems, the situation is reversed in the USPS data where BoW-S achieves in general lower test error.

## 7   Related Work

There is much previous work on a related topic of incorporating additional constraints, using prior knowledge of the problem. Shivaswamy and Jebara [24] use a geometric motivation to modify SVMs. Their effort and other related works, first deduce some additional knowledge about the problem [16, 20, 1], and keep it fixed while learning. In contrast, our method learns together the classifier and some additional information.

Another line of research is about algorithms that are maintaining a Gaussian distribution over weights, as opposed to uniform distribution as in our case, either AROW [4] in the online setting and its predecessors, or Gaussian Margin Machines (GMMs) [5] in the batch setting. Our motivation is similar to the motivation behind GMMs, yet it is different in few important aspects. (1) BoW maintains only $2d$ parameters, while GMM employs $d + d(d+1)/2$ as it maintains a full covariance matrix. (2) As a consequence, GMMs are not feasible to run on data with more than hundreds of features, which is further supported by the fact that GMMs were evaluated only on data of dimension 64 [5]. (3) We use directly a specialized PAC-Bayes bound for convex loss functions [10] while the analysis of GMMs uses a bound designed for the $0 - 1$ loss which is then further bounded. (4) The optimization problem of both versions of BoW is convex, while the optimization problem of GMMs is not convex, and it is only approximated with a convex problem. (5) Therefore, we can and do employ COMID [8] which is theoretically justified and fast in practice, while GMMs are trained using another technique with no convergence (to local minima) guarantees. (6) Conceptually, BoW maintains a compact set (box) while the set of possible weights for GMM is not compact. This allows us to extend our work to other types of sets (in progress), while its not clear how to extend the GMMs approach from Gaussian distributions to other objects.

## 8   Conclusion

We extend the commonly used linear classifiers to subsets of the class of possible classifiers, or in other words uniform distributions over weight vectors. Our learning algorithm is based on a worst-case margin minimization principle, and it benefits from strong theoretical guarantees based on tight PAC-Bayesian bounds. The empirical evaluation presented shows that our method performs favourably with respect to SVMs and AROW, and is more robust in the presence of label noise. We

plan to study the integration of kernels, extend our framework for various shapes and problems, and develop specialized large scale algorithms.

**Acknowledgments:** The paper was partially supported by an Israeli Science Foundation grant ISF-1567/10 and by a Google research award.

# References

[1] J. Bi and T. Zhang. Support vector classification with input data uncertainty. In *NIPS*, 2004.

[2] J. Blitzer, M. Dredze, and F. Pereira. Biographies, bollywood, boom-boxes and blenders: Domain adaptation for sentiment classification. In *ACL*, 2007.

[3] C. Cortes and V. Vapnik. Support-vector networks. *Machine Learning*, 20(3):273–297, September 1995.

[4] K. Crammer, A. Kulesza, and M. Dredze. Adaptive regularization of weighted vectors. In *NIPS*, 2009.

[5] K. Crammer, M. Mohri, and F. Pereira. Gaussian margin machines. In *AISTATS*, 2009.

[6] M. Dredze, K. Crammer, and F. Pereira. Confidence-weighted linear classification. In *ICML*, 2008.

[7] J. Duchi, E. Hazan, and Y. Singer. Adaptive subgradient methods for online learning and stochastic optimization. In *COLT*, 2010.

[8] J. Duchi, S. Shalev-Shwartz, Y. Singer, and A. Tewari. Composite objective mirror descent. In *COLT*, pages 250–264, 2010.

[9] Y. Freund and R.E. Schapire. A decision-theoretic generalization of on-line learning and an application to boosting. In *Euro-COLT*, pages 23–37, 1995.

[10] P. Germain, A. Lacasse, F. Laviolette, and M. Marchand. Pac-bayesian learning of linear classifiers. In *ICML*, 2009.

[11] T. Hastie, R. Tibshirani, and J. Friedman. *The Elements of Statistical Learning: Data Mining, Inference, and Prediction*. Springer, 2001.

[12] R. Herbrich, T. Graepel, and C. Campbell. Robust Bayes point machines. In *ESANN 2000*, pages 49–54, 2000.

[13] R. Herbrich, T. Graepel, and C. Campbell. Bayes point machines. *JMLR*, 1:245–279, 2001.

[14] P.J. Huber. Robust estimation of a location parameter. *Annals of Statistics*, 53:73101, 1964.

[15] T. Jaakkola and M. Jordan. A variational approach to Bayesian logistic regression models and their extensions. In *Workshop on Artificial Intelligence and Statistics*, 1997.

[16] G. Lanckriet, L. Ghaoui, C. Bhattacharyya, and M. Jordan. A robust minimax approach to classification. *JMLR*, 3:555–582, 2002.

[17] J. Langford and M. Seeger. Bounds for averaging classifiers. Technical report, CMU-CS-01-102, 2002.

[18] J. Langford and J. Shawe-Taylor. PAC-bayes and margins. In *NIPS*, 2002.

[19] D. McAllester. PAC-Bayesian model averaging. In *COLT*, 1999.

[20] J. Nath, C. Bhattacharyya, and M. Murty. Clustering based large margin classification: A scalable approach using SOCP formulation. In *KDD*, 2006.

[21] J. Pearl. *Probabilistic Reasoning in Intelligent Systems: Networks of Plausible Inference*. Morgan Kaufmann, 1988.

[22] B. Schölkopf and A. J. Smola. *Learning with Kernels: Support Vector Machines, Regularization, Optimization and Beyond*. MIT Press, 2002.

[23] M. Seeger. PAC-Bayesian generalization bounds for gaussian processes. *JMLR*, 3:233–269, 2002.

[24] P. Shivaswamy and T. Jebara. Ellipsoidal kernel machines. In *AISTATS*, 2007.

[25] V. N. Vapnik. *Statistical Learning Theory*. Wiley, 1998.

[26] M.H. Wright. The interior-point revolution in optimization: history, recent developments, and lasting consequences. *Bull. Amer. Math. Soc.*, 42:39–56, 2005.

